# Computational Elements of the Adaptive Controller of the Human Arm

**Reza Shadmehr and Ferdinando A. Mussa-Ivaldi**
Dept. of Brain and Cognitive Sciences
M. I. T., Cambridge, MA 02139
Email: reza@ai.mit.edu, sandro@ai.mit.edu

## Abstract

We consider the problem of how the CNS learns to control dynamics of a mechanical system. By using a paradigm where a subject's hand interacts with a virtual mechanical environment, we show that learning control is via composition of a model of the imposed dynamics. Some properties of the computational elements with which the CNS composes this model are inferred through the generalization capabilities of the subject outside the training data.

## 1 Introduction

At about the age of three months, children become interested in tactile exploration of objects around them. They attempt to reach for an object, but often fail to properly control their arm and end up missing their target. In the ensuing weeks, they rapidly improve and soon they can not only reach accurately, they can also pick up the object and place it. Intriguingly, during this period of learning they tend to perform rapid, flailing–like movements of their arm, as if trying to "excite" the plant that they wish to control in order to build a model of its dynamics.

From a control perspective, having a model of the arm's skeletal dynamics seems necessary because of the relatively low gain of the fast acting feedback system in the spinal neuro-muscular controllers (Crago et al. 1976), and the long delays in transmission of sensory information to the supra-spinal centers. Such a model could be used by the CNS to predict the muscular forces that need to be produced in order to move the arm along a desired trajectory. Yet, this model by itself is not sufficient

for performing a contact task because most objects which our hand interacts with change the arm's dynamics significantly. We are left with a situation in which we need to be able to quickly acquire a model of an object's dynamics so that we can incorporate it in the control system for the arm. How we learn to construct a model of a dynamical system and how our brains represent the composed model are the subjects of this research.

## 2   Learning Dynamics of a Mechanical System

To make the idea behind learning dynamics evident, consider the example of controlling a robotic arm. The arm may be seen as an inertially dominated mechanical admitance, accepting force as input and producing a change in state as its output:

$$\ddot{q} = H(q)^{-1}(F - C(q, \dot{q})) \tag{1}$$

where $q$ is the configuration of the robot, $H$ is the inertia tensor, $F$ is the input force from some controllable source (e.g., motors), and $C$ is the coriolis/centripetal forces. In learning to control the arm, i.e., having it follow a certain state trajectory or reach a final state, we form a model which has as its input the desired change in the state of the arm and receive from its output a quantity representing the force that should be produced by the actuators. Therefore, what needs to be learned is a map from state and desired changes in state to force:

$$\hat{D}(q, \dot{q}, \ddot{q}_d) = \hat{H}(q)\ddot{q}_d + \hat{C}(q, \dot{q}) \tag{2}$$

Combine the above model with a simple PD feedback system,

$$F = \hat{D} + \hat{H}K(q - q_d) + \hat{H}B(\dot{q} - \dot{q}_d)$$

and the dynamics of the system in Eq. (1) can now be written in terms of a new variable $s = q - q_d$, i.e., the error in the trajectory. It is easy to see that if we have $\hat{H} \approx H$ and $\hat{C} \approx C$, and if $K$ and $B$ are positive definite, then $s$ will be a decreasing function of time, i.e., the system will be globally stable.

Learning dynamics means forming the map in Eq. (2). The computational elements which we might use to do this may vary from simple memory cells that each have an address in the state space (e.g., Albus 1975, Raibert & Wimberly 1984, Miller et al. 1987), to locally linear functions restricted to regions where we have data (Moore & Atkeson 1994), to sigmoids (Gomi & Kawato 1990) and radial basis functions which can broadly encode the state space (Botros & Atkeson 1991). Clearly, the choice that we make in our computational elements will affect how the learned map will generalize its behavior to regions of the state space outside of the training data. Furthermore, since the task is to learn dynamics of a mechanical system (as opposed to, for example, dynamics of a financial market), certain properties of mechanical systems can be used to guide us in our choice for the computational elements. For example, the map from states to forces for any mechanical system can be linearly parameterized in terms of its mass properties (Slotine and Li, 1991). In an inertially dominated system (like a multi-joint arm) these masses may be unknown, but the fact that the dynamics can be linearized in terms of the unknowns makes the task of learning control much simpler and orders of magnitude faster than using, for example, an unstructured memory based approach.

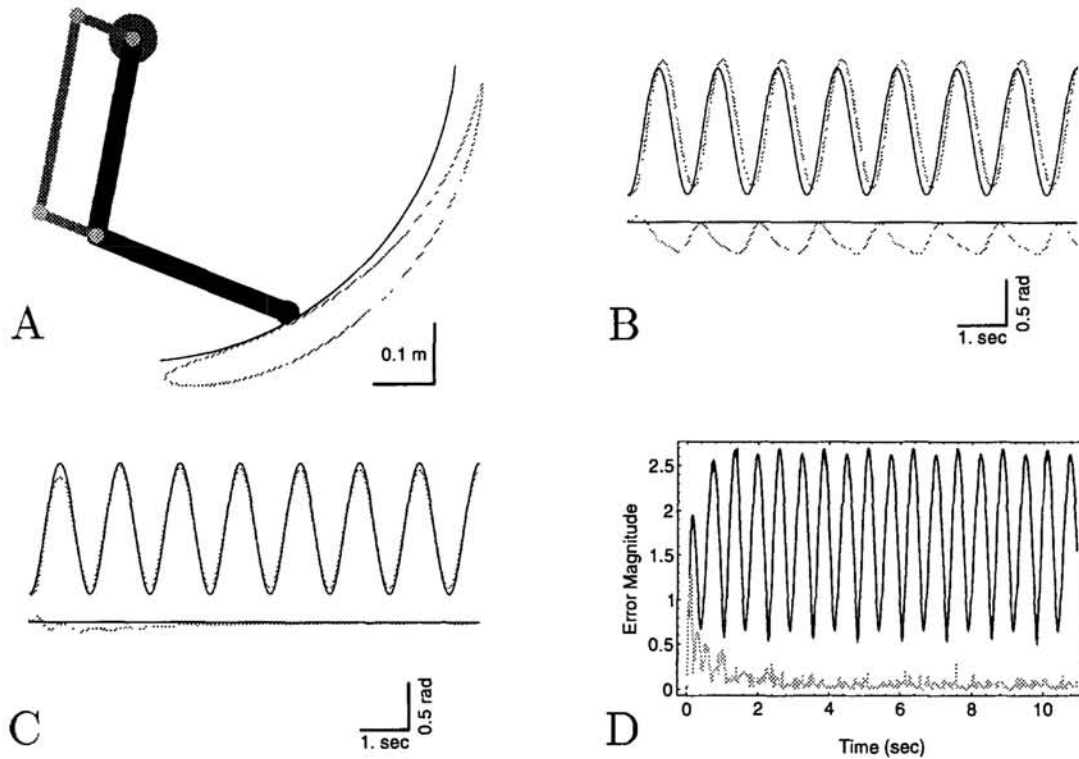

**Figure 1:** Dynamics of a real 2 DOF robot was learned so to produce a desired trajectory. **A**: Schematic of the robot. The desired trajectory is the quarter circle. Performance of a PD controller is shown by the gray line, as well as in **B**, where joint trajectories are drawn: the upper trace is the shoulder joint and the lower trace is the elbow joint. Desired joint trajectory is solid line, actual trajectory is the gray line. **C**: Performance when the PD controller is coupled with an adaptive model. **D**: Error in trajectory. Solid line is PD, Gray line is PD+adaptation.

To illustrate this point, consider the task of learning to control a real robot arm. Starting with the assumption that the plant has 2 degrees of freedom with rotational joints, inertial dynamics of Eq. (2) can be written as a product of a known matrix–function of state-dependent geometric transformations $Y$, and an unknown (but constant) vector $a$, representing the masses, center of masses, and link lengths: $\hat{D}(q, \dot{q}, \ddot{q}_d) = Y(q, \dot{q}, \ddot{q}_d)\, a$. The matrix $Y$ serves the function of referring the unknown masses to their center of rotation and is a geometric transformation which can be derived from our assumption regarding the structure of the robot. It is these geometric transformations that can guide us in choosing the computational elements for encoding the sensory data ($q$ and $\dot{q}$).

We used this approach to learn to control a real robot. The adaptation law was derived from a Lyapunov criterion, as shown by Slotine and Li (1991):

$$\dot{\hat{a}} = -Y^T(q, \dot{q}, \ddot{q}_d)\,(\dot{q} - \dot{q}_d(t) + q - q_d(t))$$

The system converged to a very low trajectory tracking error within only three periods of the movement (Fig. 1). This performance is achieved despite the fact that our model of dynamics ignores frictional forces, noise and delay in the sensors, and dynamics of the actuators. In contrast, using a sigmoid function as the basic com-

putational element of the map and training via back-propagation led to comparable levels of performance in over 4000 repetitions of the training data (Shadmehr 1990). The difference in performance of these two approaches was strictly due to the choice of the computational elements with which the map of Eq. (2) was formed.

Now consider the task of a child learning dynamics of his arm, or that of an adult picking up a hammer and pounding a nail. We can scarcely afford thousands of practice trials before we have built an adequate model of dynamics. Our proposal is that because dynamics of mechanical systems are distinctly structured, perhaps our brains also use computational elements that are particularly suited for learning dynamics of a motor task (as we did in learning to control the robot in Fig. 1). How to determine the structure of these elements is the subject of the following sections.

## 3   A Virtual Mechanical Environment

To understand how humans represent learned dynamics of a motor task, we designed a paradigm where subjects reached to a target while their hand interacted with a virtual mechanical environment. This environment was a force field produced by a manipulandum whose end-effector was grasped by the subject. The field of forces depended only on the velocity of the hand, e.g., $F = B\dot{x}$, as shown in Fig. 2A, and significantly changed the dynamics of the limb: When the robot's motors were turned off (*null field* condition), movements were smooth, straight line trajectories to the target (Fig. 2B). When coupled with the field however, the hand's trajectory was now significantly skewed from the straight line path (Fig. 2C).

It has been suggested that in making a reaching movement, the brain formulates a kinematic plan describing a straight hand path along a smooth trajectory to the target (Morasso 1981). Initially we asked whether this plan was independent of the dynamics of the moving limb. If so, as the subject practiced in the environment, the hand path should converge to the straight line, smooth trajectory observed in the null field. Indeed, with practice, trajectories in the force field did converge to those in the null field. This was quantified by a measure of correlation which for all eight subjects increased monotonically with practice time.

If the CNS adapted to the force field by composing a model of its dynamics, then removal of the field at the onset of movement (un-be-known to the subject) should lead to discrepancies between the actual field and the one predicted by the subject's model, resulting in distorted trajectories which we call *after-effects*. The expected dynamics of these after-effects can be predicted by a simple model of the upper arm (Shadmehr and Mussa-Ivaldi 1994). Since the after-effects are a by-product of the learning process, we expected that as subjects adapted to the field, their performance in the null field would gradually degrade. We observed this gradual growth of the after-effects, leading to grossly distorted trajectories in the null field after subjects had adapted to the force field (Fig. 2D). This evidence suggested that the CNS composed a model of the field and used this model to compensate for the forces which it predicted the hand would encounter during a movement.

The information contained in the learned model is a map whose input is the state and the desired change in state of the limb, and whose output is force (Eq. 2). How is this map implemented by the CNS? Let us assume that the approximation is via

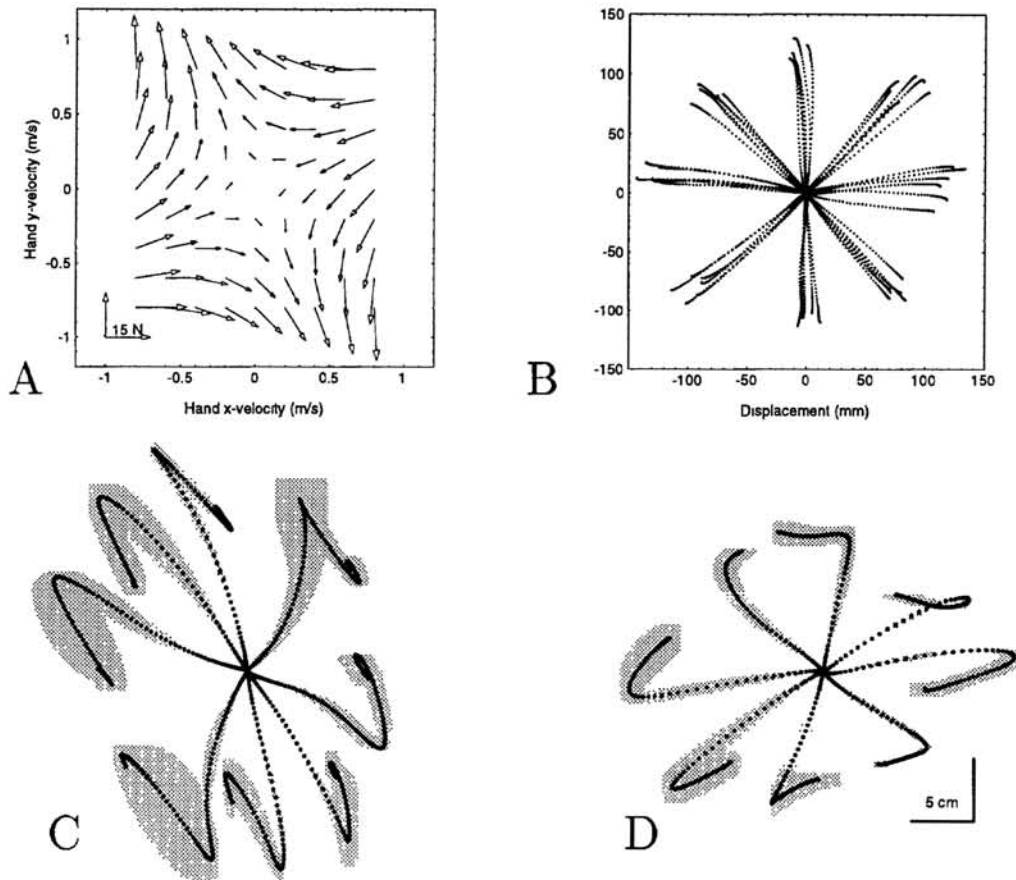

**Figure 2: A**: The virtual mechanical environment as a force field. **B**: Trajectory of reaching movements (center–out) to 8 targets in a null field. **C**: Average±standard-deviation of reaches to same targets when the field was on, before adaptation. **D**: After-affects of adaptation, i.e., when moving in a null field but expecting the field.

a distributed set of computational elements (Poggio 1990). What are the properties of these elements? An important property may be the spatial bandwidth, i.e., the size of the receptive field in the input space (the portion of the input space where the element generates a significant output). This property greatly influences how the CNS might interpolate between states which it has visited during training, and whether it can generalize to regions beyond the boundary of the training data.

For example, in eye movements, it has been suggested that a model of dynamics of the eye is stored in the cerebellum (Shidara et al. 1992). Cells which encode this model (Purkinje cells) vary their firing rate as a linear function of the state of the eye, and the sum of their outputs (firing rates) correlates well with the force that the muscles need to produce to move the eye. Therefore, the model of eye's dynamics is encoded via cells with very large receptive fields. On the other hand, cells which take part in learning a visual hyperacuity task may have very small receptive fields (Poggio et al. 1992), resulting in a situation where training in a localized region does not lead to generalization.

In learning control of our limbs, one possibility for the computational elements is the neural control circuits in the spinal cord (Mussa-Ivaldi 1992). Upon activation of

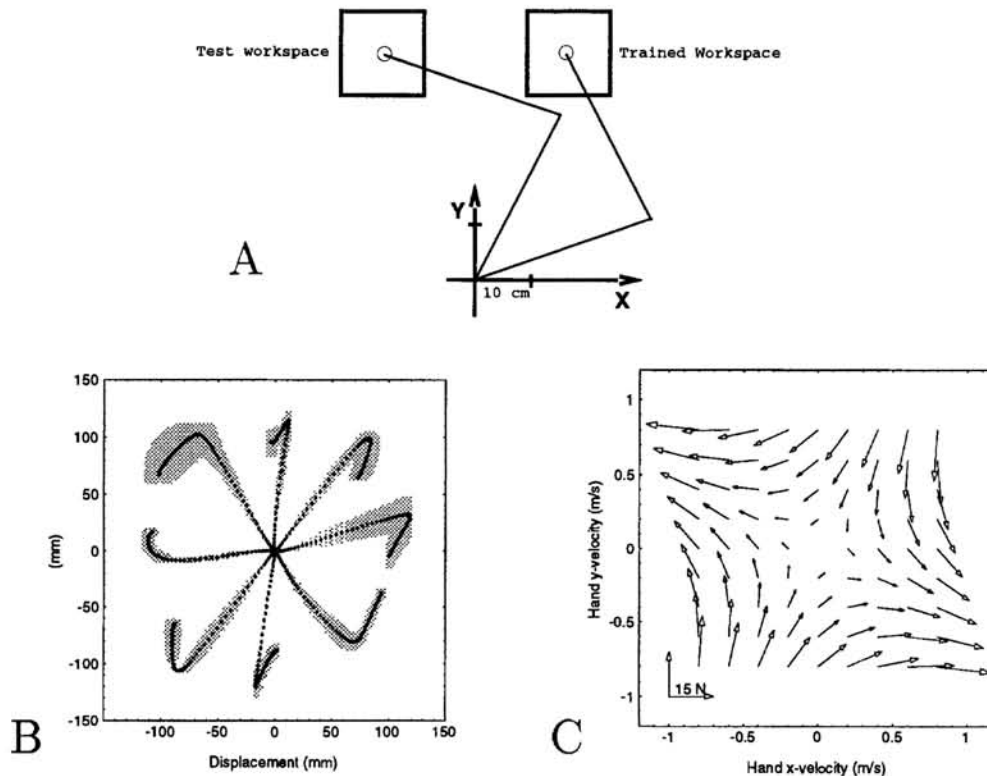

**Figure 3:** **A**: Schematic of subject's arm and the trained region of the workspace where the force field was presented and the "test" region where the transferred effects were measured. **B**: After-effects at the test region. **C**: A joint-based translation of the force field shown in Fig. 2A to the novel workspace. This is the field that the subject expected at the test region.

one such circuit, muscles produce a time varying force field, i.e., forces which depend on the state of the limb (position and velocity) and time (Mussa-Ivaldi et al. 1990). Let us call the force function produced by one such motor element $f_i(q, \dot{q}, t)$. It turns out that as one changes the amount of activation to a motor element, the output forces essentially scale. When two such motor elements are activated, the resulting force field is a linear combination of the two individual fields (Bizzi et al. 1991): $f = \sum_{i=1}^{2} c_i f_i(q, \dot{q}, t)$.

Now consider the task of learning to move in the field shown in Fig. 2A. The model that the CNS builds is a map from state of the limb to forces imposed by the environment. Following the above scenario, the task is to find coefficients $c_i$ for each element such that the output field is a good approximation of the environmental field. Unlike the computational elements of a visual task however, we may postulate that the motor elements are characterized by their broad receptive fields. This is because muscular force changes gradually as a function of the state of the limb and therefore its output force is non zero for wide region of the state space. It follows that if learning dynamics was accomplished through formation of a map whose computational elements were these motor functions, then because of the large spatial bandwidth of the elements the composed model should be able to generalize to well beyond the region of the training data.

To test this, we limited the region of the input space for which training data was provided and quantified the subject's ability to generalize to a region outside the training set. Specifically, we limited the workspace where practice movements in the force field took place and asked whether *local* exposure to the field led to after-effects in other regions (Fig. 3A). We found that local training resulted in after-effects in parts of the workspace where no exposure to the field had taken place (Fig. 3B). This indicated that the model composed by the CNS predicted specific forces well outside the region in which it had been trained. The existence of this generalization showed that the computational elements with which the internal model was implemented had broad receptive fields.

The transferred after-effects (Fig. 3B) show that at the novel region of the workspace, the subject's model of the environment predicted very different forces than the one on which the subject had been trained on (compare with Fig. 2D). This rejected the hypothesis that the composed model was a simple mapping (i.e., translation invariant) in a hand-based coordinate system, i.e., from states of the arm to forces on the hand. The alternate hypothesis was that the composed model related observed states of the arm to forces that needed to be produced by the muscles and was translation invariant in a coordinate system based on the joints and muscles. This would be the case, for example, if the computational elements encoded the state of the arm linearly (analogous to Purkinje cells for the case of eye movements) in joint space.

To test this idea, we translated the field in which the subject had practiced to the novel region in a coordinate system defined based on the joint space of the subject's arm, resulting in the field shown in Fig. 3C. We recorded the performance of the subjects in this new field at the novel region of the workspace (after they had been trained on field of Fig. 2A) and found that performance was near optimum at the first exposure. This indicated that the geometric structure of the composed model supported transfer of information in an intrinsic, e.g., joint based, coordinate system. This result is consistent with the hypothesis that the computational elements involved in this learning task broadly encode the state space and represent their input in a joint-based coordinate system and not a hand-based one.

## 4    Conclusions

In learning control of an inertially dominated mechanical system, knowledge of the system's geometric constraints can direct us to choose our computational elements such that learning is significantly faciliated. This was illustrated by an example of a real robot arm: starting with no knowledge of its dynamics, a reasonable model was learned within 3 periods of a movements (as opposed to thousands of movements when the computational elements were chosen without regard to the geometric properties). We argued that in learning to control the human arm, the CNS might also make assumption regarding geometric properties of its links and use specialized computational elements which facilitate learning of dynamics.

One possibility for these elements are the discrete neuronal circuits found in the spinal cord. The function of these circuits can be mathematically formulated such that a map representing inverse dynamics of the arm is formed via a combination of the elements. Because these computational elements encode their input space

broadly, i.e., has significant output for a wide region of the input space, we expected that if subjects learned a dynamical process from localized training data, then the formed model should generalize to novel regions of the state space. Indeed we found that the subjects transferred the training information to novel regions of the state space, and this transfer took place in a coordinate system similar to that of the joints and muscles. We therefore suggest that the CNS learns control of the arm through formation of a model whose computational elements broadly encode the state space, and that these elements may be neuronal circuits of the spinal cord.

**Acknowledgments**: Financial support was provided in part by the NIH (AR26710) and the ONR (N00014/90/J/1946). R. S. was supported by the McDonnell–Pew Center for Cognitive Neurosciences and the Center for Biological and Computational Learning.

# References

Albus JS (1975) A new approach to manipulator control: The cerebellar model articulation controller (CMAC). *Trans ASME J Dyn Syst Meas Contr* 97:220–227.

Bizzi E, Mussa-Ivaldi FA, Giszter SF (1991) Computations underlying the execution of movement: a novel biological perspective. *Science* 253:287–291.

Botros SM, Atkeson CG (1991) Generalization properties of radial basis functions. In: Lippmann et al., *Adv. in Neural Informational Processing Systems* 3:707–713.

Crago, Houk JC, Hasan Z (1976) Regulatory actions of human stretch reflex. J Neurophysiol 39:5–19.

Gomi H, Kawato M (1990) Learning control for a closed loop system using feedback error learning. *Proc IEEE Conf Decision Contr.*

Miller WT, Glanz FH, Kraft LG (1987) Application of a general learning algorithm to the control of robotic manipulators. *Int J Robotics Res* 6(2):84–98.

Moore AW, Atkeson CG (1994) An investigation of memory-based function approximators for learning control. *Machine Learning*, submitted.

Mussa-Ivaldi FA, Giszter SF (1992) Vector field approximation: a computational paradigm for motor control and learning. *Biol Cybern* 67:491–500.

Mussa-Ivaldi FA, Giszter SF, Bizzi E (1990) Motor-space coding in the central nervous system. *Cold Spring Harbor Symp Quant Biol* 55:827–835.

Poggio T (1990) A theory of how the brain might work. *Cold Spring Harbor Symp Quant Biol* 55:899–910.

Poggio T, Fahle M, Edelman S (1992) Fast perceptual learning in visual hyperacuity. *Science* 256:1018–1021.

Raibert MH, Wimberly FC (1984) Tabular control of balance in a dynamic legged system. *IEEE Trans Systems, Man, Cybernetics* SMC-14(2):334–339.

Shadmehr R (1990) Learning virtual equilibrium trajectories for control of a robot arm. *Neural Computation* 2:436–446.

Shadmehr R, Mussa-Ivaldi FA (1994) Adaptive representation of dynamics during learning of a motor task. *J Neuroscience*, in press.

Shidara M, Kawano K, Gomi H, Kawato M (1993) Inverse-dynamics model eye movement control by Purkinje cells in the cerebellum. *Nature* 365:50–52. Slotine JJE, Li W (1991) *Applied Nonlinear Control*, Prentice Hall, Englewood Cliffs, New Jersey.
